# A Model of Recurrent Interactions in Primary Visual Cortex

**Emanuel Todorov, Athanassios Siapas and David Somers**
Dept. of Brain and Cognitive Sciences
E25-526, MIT, Cambridge, MA 02139
Email: {emo,thanos,somers}@ai.mit.edu

## Abstract

A general feature of the cerebral cortex is its massive interconnectivity - it has been estimated anatomically [19] that cortical neurons receive upwards of 5,000 synapses, the majority of which originate from other nearby cortical neurons. Numerous experiments in primary visual cortex (V1) have revealed strongly nonlinear interactions between stimulus elements which activate classical and non-classical receptive field regions. Recurrent cortical connections likely contribute substantially to these effects. However, most theories of visual processing have either assumed a feedforward processing scheme [7], or have used recurrent interactions to account for isolated effects only [1, 16, 18]. Since nonlinear systems cannot in general be taken apart and analyzed in pieces, it is not clear what one learns by building a recurrent model that only accounts for one, or very few phenomena. Here we develop a relatively simple model of recurrent interactions in V1, that reflects major anatomical and physiological features of intracortical connectivity, and *simultaneously* accounts for a wide range of phenomena observed physiologically. All phenomena we address are strongly nonlinear, and cannot be explained by linear feedforward models.

## 1 The Model

We analyze the mean firing rates observed in oriented V1 cells in response to stimuli consisting of an inner circular grating and an outer annular grating. Mean responses of individual cells are modeled by single-valued "cellular" response functions, whose arguments are the mean firing rates of the cell's inputs and their maximal synaptic conductances.

## 1.1 Neuronal model

Each neuron is modeled as a single voltage compartment in which the membrane potential $V$ is given by:

$$C_m \frac{dV(t)}{dt} = g_{\text{ex}}(t)(E_{\text{ex}} - V(t)) + g_{\text{inh}}(t)(E_{\text{inh}} - V(t)) +$$
$$g_{\text{leak}}(E_{\text{leak}} - V(t)) + g_{\text{ahp}}(t)(E_{\text{ahp}} - V(t))$$

where $C_m$ is the membrane capacitance, $E_x$ is the reversal potential for current $x$, and $g_x$ is the conductance for that current. If the voltage exceeds a threshold $V_\theta$, a spike is generated, and afterhyperpolarizing currents are activated. The conductances for excitatory and inhibitory currents are modeled as sums of $\alpha$-functions, and the *ahp* conductance is modeled as a decaying exponential. The model consists of two distinct cell types, excitatory and inhibitory, with realistic cellular parameters [13], similar to the ones used in [17]. To compute the response functions for the two cell types, we simulated one cell of each type, receiving excitatory and inhibitory Poisson inputs. The synaptic strengths were held constant, while the rates of the excitatory and inhibitory inputs were varied independently.

Although the driving forces for excitation and inhibition vary, we found that single cell responses can be accurately modeled if incoming excitation and inhibition are combined linearly, and the net input is passed through a response function that is approximately threshold-linear, with some smoothing around threshold. This is consistent with the results of intracellular experiments that show linear synaptic interactions in visual cortex[5]. Note that the cellular functions are not sigmoids, and thus saturating responses could be achieved only through network interactions.

## 1.2 Cortical connectivity

The visual cortex shares with many other cortical areas a similar pattern of intra-areal connections [12]. Excitatory cells make dense local projections, as well as long-range horizontal projections that usually contact cells with similar response properties. Inhibitory cells make only local projections, which are spread further in space than the local excitatory connections [10]. We assume that cells with similar response properties have a higher probability of connection, and that probability falls down with distance in "feature" space. For simplicity, we consider only two feature dimensions: orientation and RF center in visual space. Since we are dealing with stimuli with radial symmetry, one spatial dimension is sufficient. The extension to more dimensions, i.e. another spatial dimension, direction selectivity, ocularity, etc., is straightforward.

We assume that the feature space is filled uniformly with excitatory and inhibitory cells. Rather than modeling individual cells, we model a grid of locations, and for each location we compute the mean firing rate of cells present there. The connectivity is defined by two projection kernels $K_{ex}, K_{in}$ (one for each presynaptic cell type) and weights $W_{ee}, W_{ei}, W_{ie}, W_{ii}$, corresponding to the number and strength of synapses made onto excitatory/inhibitory cells. The excitatory projection has sharper tuning in orientation space, and bigger spread in visual space (Figure 1).

## 1.3 Visual stimuli and thalamocortical input

The visual stimulus is defined by five parameters: diameter $d$ of the inner grating (the outer is assumed infinite), log contrast $c_1, c_2$, and orientation $\theta_1, \theta_2$ of each grating. The two gratings are always centered in the spatial center of the model.

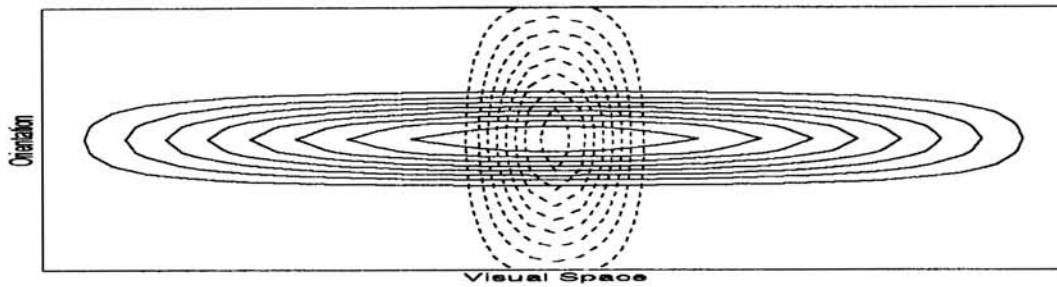

Figure 1: Excitatory (solid) and Inhibitory (dashed) connectivity kernels.

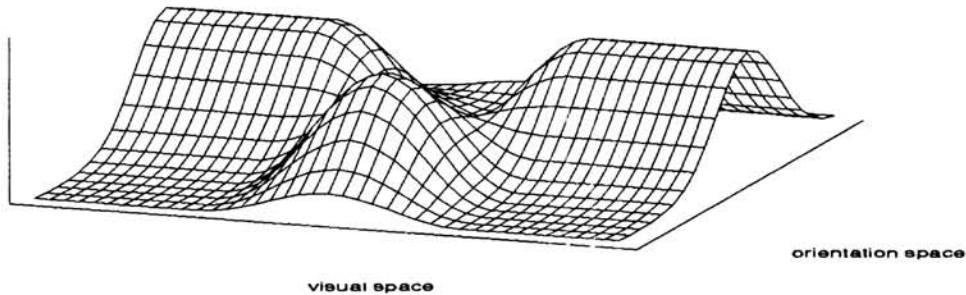

Figure 2: LGN input for a stimulus with high contrast, orthogonal orientations of center and surround gratings.

Each cortical cell receives LGN input which is the product of log contrast, orientation tuning, and convolution of the stimulus with a spatial receptive field. The LGN input computed in this way is multiplied by $LGN_{ex}, LGN_{in}$ and sent to the cortical cells. Figure 2 shows an example of what the $LGN$ input looks like.

## 2 Results

For given input LGN input, we computed the steady-state activity in cortex iteratively (about 30 iteration were required). Since we studied the model for gradually changing stimulus parameters, it was possible to use the solution for one set of parameters as an initial guess for the next solution, which resulted in significant speedup. The results presented here are for the excitatory population, since i) it provides the output of the cortex; ii) contains four times more cells than the inhibitory population, and therefore is more likely to be recorded from. All results were obtained for the same set of parameters.

### 2.1 Classical RF effects

First we simulate responses to a central grating ($1deg$ diameter) for increasing log-contrast levels. It has been repeatedly observed that although LGN firing increases linearly with stimulus log-contrast, the contrast response functions observed in V1 saturate, and may even supersaturate or decline[11, 2]. The most complete and recent model of that phenomena [6, 3] assumes shunting inhibition, which contradicts recent intracellular observations [4, 5]. Furthermore, that model cannot explain supersaturation. Our model achieves saturation (Figure 3A) for high contrast levels,

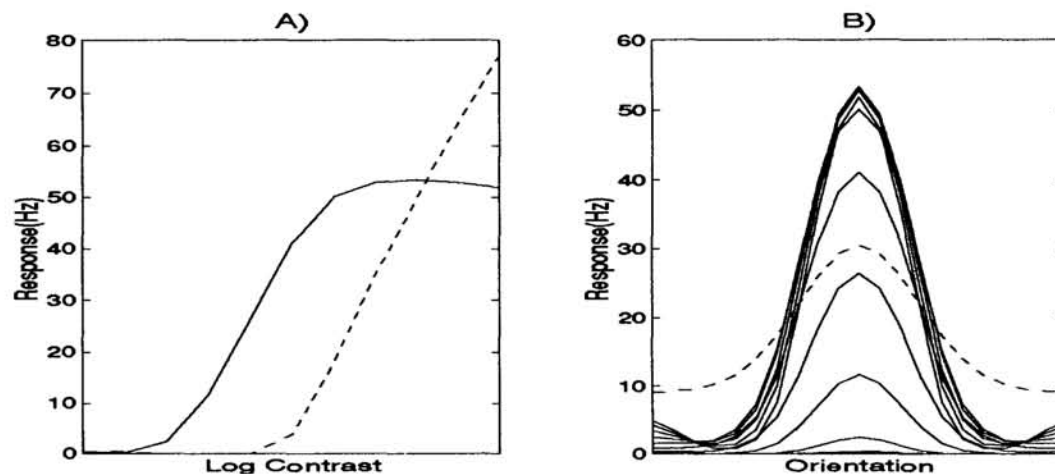

Figure 3: Classical RF effects. A) Contrast response function for excitatory (solid) and inhibitory (dashed) cells. B) Orientation tuning for different contrast levels (solid); scaled LGN input (dashed).

and can easily achieve supersaturation for different parameter setting. Instead of using shunting (divisive) inhibition, we suggest that inhibitory neurons have higher contrast thresholds than excitatory neurons, i.e. the direct LGN input to inhibitory cells is weaker. Note that we only need a subpopulation of inhibitory neurons with that property; the rest can have the same response threshold as the excitatory population.

Another well-known property of V1 neurons is that their orientation tuning is roughly contrast-invariant[14]. The LGN input tuning is invariant, therefore this property is easily obtained in models where V1 responses are almost linear, rather than saturating [1, 16]. Achieving both contrast-invariant tuning and contrast saturation for the entire population (while singe cell feedforward response functions are non-saturating) is non-trivial. The problem is that invariant tuning requires the responses at all orientations saturate simultaneously. This is the case in our model (Figure 3B) - we found that the tuning (half width at half amplitude) varied within a 5deg range as contrast increased.

## 2.2   Extraclassical RF effects

Next we consider stimuli which include both a center and a surround grating. In the first set of simulations we held the diameter constant (at 1deg) and varied stimulus contrast and orientation. It has been observed [9, 15] that a high contrast iso-orientation surround stimulus facilitates responses to a low contrast, but suppresses responses to a high contrast center stimulus. This behavior is captured very well by our model (Figure 4A). The strong response to the surround stimulus alone is partially due to direct thalamic input (i.e. the thalamocortical projective field is larger than the classical receptive field of a V1 cell). The response to an orthogonal surround is between the center and iso-orientation surround responses, as observed in [9].

Many neurons in V1 respond optimally to a center grating with a certain diameter, but their response decreases as the diameter increases (end-stopping). End-stopping in the model is shown in Figure 4B - responses to increasing grating diameter reach a peak and then decrease. In experiments it has been observed that the

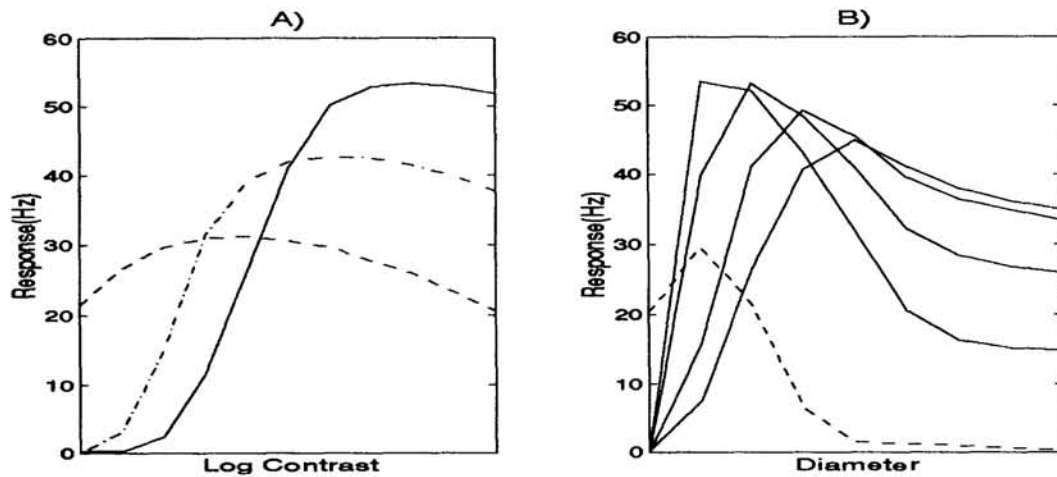

Figure 4: Extraclassical RF effects. A) Contrast response functions for center (solid), center + iso–orientation surround (dashed), center + orthogonal surround (dash-dot). B) Length tuning for 4 center log contrast levels (1, .75, .5, .4), response to surround of high contrast (dashed).

border between the excitatory and inhibitory regions shifts outward (rightward) as stimulus contrast levels decline[8]. Our model also achieves this effect (for other parameter settings it can shift more downward than rightward). Note also that a center grating with maximal contrast reaches its peak response for a diameter value which is 3 times smaller than the diameter for which responses to a surround grating disappear. This is interesting because both the peak response to a central grating, and the first response to a surround grating can be used to define the extent of the classical RF - in this case clearly leading to very different definitions. This effect (shown in Figure 4B) has also been recently observed in V1 [15].

## 3  Population Modeling and Variability

So far we have analyzed the population firing rates in the model, and compared them to physiological observations. Unfortunately, in many cases the limited sample size, or the variability in a given physiological experiment does not allow an accurate estimate of what the population response might be. In such cases researchers only describe individual cells, which are not necessarily representative. How can findings reported in this way be captured in a population model? The parameters of the model could be modified in order capture the behavior of individual cells on the population level; or, further subdivisions into neuronal subpopulations may be introduced explicitly. The approach we prefer is to increase the variance of the number of connections made across individual cells, while maintaining the same mean values. We consider variations in the amount of excitatory and inhibitory synapses that a particular cell receives from other cortical neurons. Note that the presynaptic origin of these inputs is still the "average" cortical cell , and is not chosen from a subpopulation with special response properties.

The two examples we show have recently been reported in [15]. If we increase the cortical input (excitation by 100%, inhibition by 30%), we obtain a completely patch-suppressed cell, i.e. it does not respond to a center + iso–orientation surround stimulus - Figure 5A. Figure 5B shows that this cell is an "orientation contrast detector", i.e. it responds well to $0deg$ center + $90deg$ surround, and to $90deg$

center + 0*deg* surround. Interestingly, the cells with that property reported in [15] were all patch-suppressed. Note also that our cell has a supersaturating center response - we found that it always accompanies patch suppression in the model.

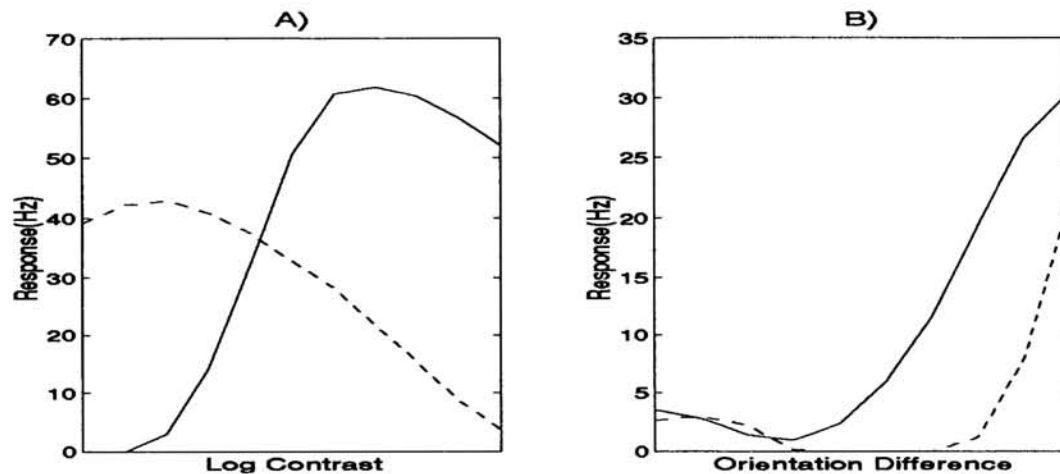

Figure 5: Orientation discontinuity detection for a strongly patch- suppressed cell. A) Contrast response functions for center (solid) and center + iso–orientation surround (dashed). B) Cell's response for 0*deg* center, 0 − 90*deg* surround (solid) and 90*deg* center, 90 − 0*deg* surround.

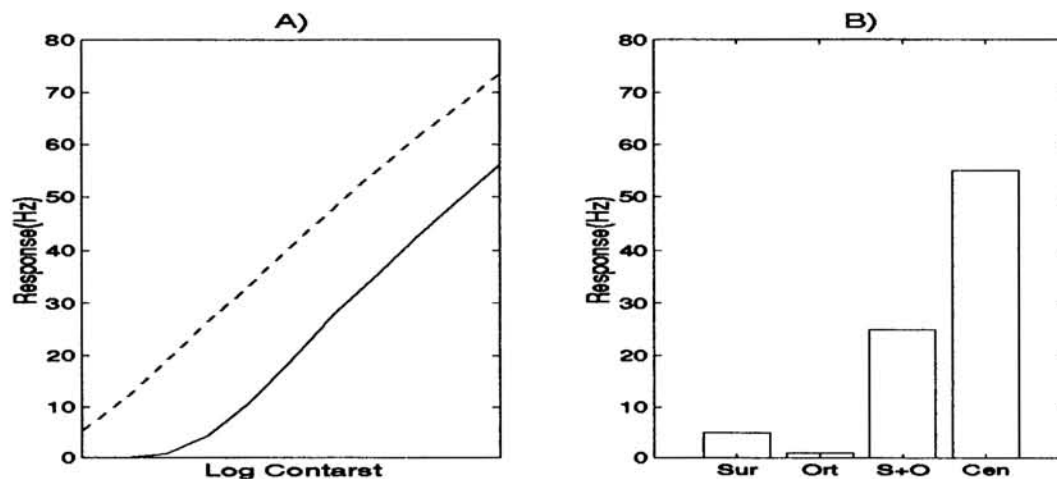

Figure 6: Nonlinear summation for a non patch-suppressed cell. A) Contrast response functions for center (solid) and center + iso–orientation surround (dashed). B) Cell's response for iso–orientation surround, orthogonal center, surround + center, center only.

The second example is a cell receiving 15% of the average cortical input. Not surprisingly, its contrast response function does not saturate - Figure 6A. However, this cell exhibits an interesting nonlinear property - it respond well to a combination of an iso–orientation surround + orthogonal center, but does not respond to either stimulus alone (Figure 6B). It is not clear from [15] whether the cells with this property had saturating contrast response functions.

## 4  Conclusion

We presented a model of recurrent computation in primary visual cortex that relies on a limited set of physiologically plausible mechanisms. In particular, we used the different cellular properties of excitatory and inhibitory neurons and the non-isotropic shape of lateral connections (Figure 1). Due to space limitations, we only presented simulation results, rather than analyzing the effects of specific parameters. A preliminary version of such analysis is given in [17] for local effects, and will be developed elsewhere for lateral interaction effects.

Our goal here was to propose a framework for studying recurrent computation in V1, that is relatively simple, yet rich enough to simultaneously account for a wide range of physiological observations. Such a framework is needed if we are to analyse systematically the fundamental role of recurrent interactions in neocortex.

## References

[1] Ben-Yishai, R., Lev Bar-Or, R. & Sompolinsky, H. *Proc. Natl. Acad. Sci. U.S.A.* **92**, 3844-3848 (1995).

[2] Bonds, A.B. *Visual Neurosci.* **6**, 239-255 (1991).

[3] Carandini, M. & Heeger, D.J. *Science,* **264**, 1333-1336 (1994).

[4] Douglas R.J., Martin K.C., Whitteridge D. An intracellular analysis of the visual responses on neurones in cat visual cortex. *J Physiology* **44**, 659-696, 1991.

[5] Ferster,D & Jagadeesh, B. *J. Neurosci.* **12**, 1262-1274(1992).

[6] Heeger, D.J. *Visual Neurosci.* **70**, 181-197 (1992).

[7] Hubel, D.H. & Wiesel, T.N. *J. Neurophysiol.* **148**, 574-591 (1959).

[8] Jagadeesh, B. & Ferster, D. *Soc Neursci Abstr.* **16** 130.11. (1990).

[9] Knierim, J.J. & Van Essen, D.C. *J. Neurophysiol.* **67**, 961-980 (1992).

[10] Kisvarday, Z.F., Martin, K.A.C., Freund, T.F., Magloczky, Z.F., Whitteridge, D., and Somogyi, D. *Exp. Brain Res.* **64**, 541-552.

[11] Li, C.Y. & Creutzfeldt, O.D. *Pflugers Arch.* **401**, 304-314 (1984).

[12] Lund J.S., Yoshioka T., Levitt, J.B. *Cereb. Cortex* **3**, 148-162.

[13] McCormick, D.A., Connors, B.W., Lighthall, J.W. & Prince, D.A. *J. Neurophysiol.* **54**, 782-806 (1985).

[14] Sclar, G. & Freeman, R.D. *Exp. Brain Res.* **46**, 457-461.

[15] Sillito, A.M., Grieve, K.L., Jones, H.E., Cudeiro, J., & Davis, J. *Nature*, Nov 1995.

[16] Somers, D.C., Nelson, S.B. & Sur, M. *J. Neurosci.* **15**, 5448-5465 (1995).

[17] Siapas A, Todorov E, Somers D. Computing the mean firing rates of ensembles of realistic neurons. *Soc Neuroscience Abstract*, 1995.

[18] Stemmler, M., Usher, M. & Niebur, E *Science* **269**, 1877-1880, (1995).

[19] White, E.L. *Cortical Circuits* 46-82 (Birkhauser, Boston, 1989).
